# Distributionally Robust Markov Decision Processes

**Huan Xu**
ECE, University of Texas at Austin
huan.xu@mail.utexas.edu

**Shie Mannor**
Department of Electrical Engineering, Technion, Israel
shie@ee.technion.ac.il

## Abstract

We consider Markov decision processes where the values of the parameters are uncertain. This uncertainty is described by a sequence of nested sets (that is, each set contains the previous one), each of which corresponds to a probabilistic guarantee for a different confidence level so that a set of admissible probability distributions of the unknown parameters is specified. This formulation models the case where the decision maker is aware of and wants to exploit some (yet imprecise) a-priori information of the distribution of parameters, and arises naturally in practice where methods to estimate the confidence region of parameters abound. We propose a decision criterion based on *distributional robustness*: the optimal policy maximizes the expected total reward under the most adversarial probability distribution over realizations of the uncertain parameters that is admissible (i.e., it agrees with the a-priori information). We show that finding the optimal distributionally robust policy can be reduced to a standard robust MDP where the parameters belong to a *single* uncertainty set, hence it can be computed in polynomial time under mild technical conditions.

## 1 Introduction

Sequential decision making in stochastic dynamic environments, also called the "planning problem," is often modeled using a Markov Decision Process (MDP, cf [1, 2, 3]). In practice, *parameter uncertainty* – the deviation of the model parameters from the true ones (reward $r$ and transition probability $p$) – often causes the performance of "optimal" policies to degrade significantly [4]. Many efforts have been made to reduce such performance variation under the robust MDP framework (e.g., [5, 6, 7, 8, 9, 10]). In this context, it is assumed that the parameters can be any member of a known set (termed the *uncertainty set*), and solutions are ranked based on their performance under the (respective) worst parameter realizations.

In this paper we extend the robust MDP framework to deal with probabilistic information on uncertain parameters. To motivate the problem, let us consider the following example. Suppose that an agent (car, plane, robot etc) wants to find a fastest path from the source location to the destination. If the passing time to area $\mathfrak{A}$ is uncertain and can be very large, then the solution to robust MDP would tend to take a detour and avoid $\mathfrak{A}$. However, if it is further known that the passing time can be large only when some unusual event (whose chance is less than, say, $10\%$), such as a storm, happens, and otherwise the passing time is reasonable, then avoiding $\mathfrak{A}$ may be overly pessimistic. The statement "the probability of the (uncertain) passing time being large is at most $10\%$" is important, and should be incorporated into the decision making paradigm. Indeed, it was observed that since the robust MDP framework ignores probabilistic information, it can provide conservative solutions [11, 12].

A different approach to embedding prior information is by adopting a Bayesian perspective on the parameters of the problem; see [11] and references therein. However, a complete Bayesian prior to the model parameters may be difficult to conjure as the decision maker may not have a reliable generative model to the uncertainty. For example, in the path planning problem above, the decision maker may not know how to assign probabilities to the model dynamics when a storm occurs. Our

approach offers a middle ground between the fully Bayesian approach and the robust approach: we want the decision maker to be able to use prior information but we do not require a complete Bayesian interpretation.

We adapt the distributionally robust approach to MDPs under parameter uncertainty. The distributionally robust formulation has been extensively studied and broadly applied in *single stage* optimization problems to effectively incorporates a-priori probabilistic information of the unknown parameters (e.g., [13, 14, 15, 16, 17, 18]). In this framework, the uncertain parameters are regarded as stochastic, with a distribution $\mu$ that is not precisely observed, yet assumed to belong to an a-priori known set $\mathcal{C}$. The objective is then formulated based on the worst-case analysis over distributions in $\mathcal{C}$. That is, given a utility function $u(x, \xi)$ where $x \in \mathcal{X}$ is the optimizing variable and $\xi$ is the unknown parameter, distributionally robust optimization solves $\max_{x \in \mathcal{X}} \left[ \inf_{\mu \in \mathcal{C}} \mathbb{E}_{\xi \sim \mu} u(x, \xi) \right]$. Indeed, such approach has also been developed in the mathematical finance community, usually in the static setup [19, 20]. Here the goal is to optimize a so-called *coherent risk measure*, which is shown to be equivalent to a distributionally robust formulation.

From a decision theory perspective, the distributionally robust approach coincides with the celebrated MaxMin Expected Utility framework [21, 22], which states that if a preference relationship among actions satisfies certain axioms, then the optimal action maximizes the minimal expected utility with respect to a class of distributions. This approach addresses the famous *neglect of probability cognitive bias* [23], i.e., the tendency to completely disregard probability when making a decision under uncertainty. Two extreme cases of such biases are the *normalcy bias*, which roughly speaking, can be states as "since a disaster has never occurred then it never will occur," and the *zero-risk bias*, which stands for the tendency of individuals to prefer small benefits that are certain to large ones that are uncertain, regardless of the size of the "certain" benefit and the expected magnitude of the uncertain one. It is easy to see that the nominal approach and the robust approach suffers from normalcy bias and zero-risk bias, respectively.

We formulate and solve the distributionally robust MDP with respect to the *nested uncertainty set*. The nesting structure implies that there are $n$ different levels of estimation, that is, $\mathcal{C}_s^1 \subseteq \mathcal{C}_s^2 \subseteq \cdots \mathcal{C}_s^n$, representing the possible parameters of the problem. The probability that the parameters of state $s$ belong to $\mathcal{C}_s^i$ is at least $\lambda_i$. We also require the parameters to be state-wise independent (i.e., the uncertainty set is a product set over states). Policies are then ranked based on their expected performance under the (respective) most adversarial distribution. The main contribution of this paper is showing that for both the finite horizon case and the discounted reward infinite horizon case, such optimal policy satisfies a Bellman type equation, and can be solved via backward iteration.

**Motivating example.** The nested-set formulation is motivated by the "multi-scenario" setup, where in different scenarios the parameters are subject to different levels of uncertainty. For instance, in the path planning example, the uncertainty of the passing time of $\mathfrak{A}$ can be modeled as a nested-set with two uncertainty sets: the parameters with at least $90\%$ belong to a small uncertainty set corresponding to "no storm," and guaranteed to belong to a large worst-case uncertainty set representing "storm" with probability of at most $10\%$. In fact the multi-layer formulations allows the decision maker to handle more than two scenarios. For example, a plane can encounter scenarios such as "normal," "storm," "big storm," and even "volcano ashes," each corresponding to a different level of parameter uncertainty. One appealing advantage of the nested-set formulation is that it does not require a precise description of the uncertainty, which leads to considerable flexibility. For example, if the uncertainty set of a robust MDP is not precisely known, then one can instead solve distributionally robust MDP with a 2-set formulation where the inner and the outer sets represent, respectively, an "optimistic" estimation and a "conservative" estimation. Additionally, the nested-set formulation also results from estimating the distributions of parameters via sampling. Such estimation is often imprecise especially when only a small number of samples is available. Instead, estimating uncertainty sets with high confidence can be made more accurate, and one can easily sharpen the approximation by incorporating more layers of confidence sets (i.e, increase $n$).

## 2 Preliminaries and Problem Setup

A (finite) MDP is defined as a 6-tuple $< T, \gamma, S, A_s, \mathbf{p}, \mathbf{r} >$ where: $T$ is the possibly infinite decision horizon; $\gamma \in (0, 1]$ is the discount factor; $S$ is the finite state set; $A_s$ is the finite action set of state $s$; $\mathbf{p}$ is the transition probability; and $\mathbf{r}$ is the expected reward. That is, for $s \in S$ and $a \in A_s$, $r(s, a)$

is the expected reward and $p(s'|s,a)$ is the probability to reach state $s'$. Following Puterman [1], we denote the set of all history-dependent randomized policies by $\Pi^{HR}$, and the set of all Markovian randomized policies by $\Pi^{MR}$. We use subscript $s$ to denote the value associated with state $s$, e.g., $\mathbf{r}_s$ denotes the vector form of rewards associated with state $s$, and $\pi_s$ is the (randomized) action chosen at state $s$ for policy $\pi$. The elements in vector $\mathbf{p}_s$ are listed in the following way: the transition probabilities of the same action are arranged in the same block, and inside each block they are listed according to the order of the next state. We use $\underline{s}$ to denote the (random) state following $s$, and $\Delta(s)$ to denote the probability simplex on $A_s$. We use $\otimes$ to represent cartesian product, e.g., $\mathbf{p} = \bigotimes_{s \in S} \mathbf{p}_s$. For a policy $\pi$, we denote the expected (discounted) total-reward under parameters $\mathbf{p}$, $\mathbf{r}$ by $u(\pi, \mathbf{p}, \mathbf{r})$, that is,

$$u(\pi, \mathbf{p}, \mathbf{r}) \triangleq \mathbb{E}_\pi^{\mathbf{p}} \{ \sum_{i=1}^{T} \gamma^{i-1} r(s_i, a_i) \}.$$

In this paper we propose and solve *distributionally robust* policy under parameter uncertainty, which incorporates a-prior information of how parameters are distributed. Suppose it is known that $\mathbf{p}$ and $\mathbf{r}$ follows some unknown distribution $\mu$ that belongs to a set $\mathcal{C}_S$. We evaluate each policy by its expected performance under the (respective) most adversarial distribution of the uncertain parameters, and a distributionally robust policy is the optimal policy according to this measure.

**Definition 1.** *A policy $\pi^* \in \pi^{HR}$ is distributionally robust* with respect to $\mathcal{C}_S$ if it satisfies that for all $\pi \in \Pi^{HR}$,

$$\inf_{\mu \in \mathcal{C}_S} \int u(\pi, \mathbf{p}, \mathbf{r}) \, d\mu(\mathbf{p}, \mathbf{r}) \leq \inf_{\mu' \in \mathcal{C}_S} \int u(\pi^*, \mathbf{p}, \mathbf{r}) \, d\mu'(\mathbf{p}, \mathbf{r}).$$

Next we specify the set of admissible distributions of uncertain parameters $\mathcal{C}_S$ investigated in this paper. Let $0 = \lambda_0 \leq \lambda_1 \leq \lambda_2 \leq \cdots \leq \lambda_n = 1$, and $\mathcal{P}_s^1 \subseteq \mathcal{P}_s^2 \subseteq \cdots \subseteq \mathcal{P}_s^n$ for $s \in S$. We use the following set of distributions $\mathcal{C}_S$ for our model.

$$\mathcal{C}_S \triangleq \{ \mu | \mu = \bigotimes_{s \in S} \mu_s; \ \mu_s \in \mathcal{C}_s, \forall s \in S \},$$

$$\text{where:} \mathcal{C}_s \triangleq \{ \mu_s | \mu_s(\mathcal{P}_s^n) = 1; \ \mu_s(\mathcal{P}_s^i) \geq \lambda_i, \ i = 1, \cdots, n-1 \}. \tag{1}$$

We briefly explain this set of distributions. For a state $s$, the condition $\mu_s(\mathcal{P}_s^n) = 1$ means that the unknown parameters $(\mathbf{p}_s, \mathbf{r}_s)$ are restricted to the outermost uncertainty set; and the condition $\mu_s(\mathcal{P}_s^i) \geq \lambda_i$ means that with probability at least $\lambda_i$, $(\mathbf{p}_s, \mathbf{r}_s) \in \mathcal{P}_s^i$. Thus, $\mathcal{P}_s^1, \cdots, \mathcal{P}_s^n$ provides probabilistic guarantees of $(\mathbf{p}_s, \mathbf{r}_s)$ for $n$ different uncertainty sets (or equivalently confidence levels). Note that $\bigotimes_{s \in S} \mu_s$ stands for the product measure generated by $\mu_s$, which indicates that the parameters among different states are independent. Throughout this paper we make a standard assumption (cf [5, 6, 8]) that $\mathcal{P}_s^i$ is nonempty, convex and compact.

## 3 Distributionally robust MDPs: The finite-horizon case.

In this section we show how to solve distributionally robust policies to MDPs having finitely many decision stages. We assume that when a state is visited multiple times, each time it can take a different parameter realization (*non-stationary model*). Equivalently, this means that multiple visits to a state can be treated as visiting different states, which leads to the Assumption 1 without loss of generality (by adding dummy states). Thus, we can partition $S$ according to the stage each state belongs to, and let $S_t$ be the set of states belong to $t^{th}$ stage. The non-stationary model is proposed in [5] because the stationary model is generally intractable and a lower-bound on it is given by the non-stationary model.

**Assumption 1.** *(i) Each state belongs to only one stage; (ii) the terminal reward equals zero; and (iii) the first stage only contains one state $s^{ini}$.*

We next define *sequentially robust policies* through a backward induction as a policy that is robust in every step for a *standard* robust MDP. We will later shows that sequentially robust policies are also distributionally robust by choosing the uncertainty set of the robust MDP carefully.

**Definition 2.** *Let $T < \infty$ and let $\mathcal{P}_s$ be the uncertainty set of state $s$. Define the following:*

1. *For $s \in S_T$, the* sequentially robust value $\tilde{v}_T(s) \triangleq 0$.

2. *For $s \in S_t$ where $t < T$, the* sequentially robust value $\tilde{v}_t(s)$ *and* sequentially robust action $\tilde{\pi}_s$ *are defined as*

$$\tilde{v}_t(s) \triangleq \max_{\pi_{\mathbf{s}} \in \Delta(s)} \left\{ \min_{(\mathbf{p}_s, \mathbf{r}_s) \in P_s} \mathbb{E}_{\pi_s}^{\mathbf{P}_s}[r(s,a) + \gamma \tilde{v}_{t+1}(\underline{s})] \right\}.$$

$$\tilde{\pi}_s \in \arg \max_{\pi_{\mathbf{s}} \in \Delta(s)} \left\{ \min_{(\mathbf{p}_s, \mathbf{r}_s) \in P_s} \mathbb{E}_{\pi_s}^{\mathbf{P}_s}[r(s,a) + \gamma \tilde{v}_{t+1}(\underline{s})] \right\}.$$

3. *A policy $\tilde{\pi}^*$ is a* sequentially robust policy *w.r.t. $P_s$ if $\forall s \in S$, $\tilde{\pi}_s^*$ is a sequentially robust action.*

A standard game theoretic argument implies that sequentially robust actions, and hence sequentially robust policies, exist. Indeed, from the literature in robust MDP (cf [5, 7, 8]) it is easy to see that a sequentially robust policy is the solution to the robust MDP where the uncertainty set is $\bigotimes_s \mathcal{P}_s$. The following theorem, which is the main result of this paper, shows that any sequentially robust policy (w.r.t. a specific uncertainty set) $\pi^*$ is distributionally robust.

**Theorem 1.** *Let $T < \infty$. Let Assumption 1 hold, and suppose that $\pi^*$ is a sequentially robust policy w.r.t. $\bigotimes_s \hat{\mathcal{P}}_s$, where*

$$\hat{\mathcal{P}}_s = \{\sum_{i=1}^n (\lambda_i - \lambda_{i-1})(\mathbf{r}_s(i), \mathbf{p}_s(i)) | (\mathbf{p}_s(i), \mathbf{r}_s(i)) \in \mathcal{P}_s^i\}.$$

*Then*

1. *$\pi^*$ is a distributionally robust policy with respect to $\mathcal{C}_s$; and*

2. *there exists $\mu^* \in \mathcal{C}_s$ such that $(\pi^*, \mu^*)$ is a saddle point. That is,*

$$\sup_{\pi \in \Pi^{HR}} \int u(\pi, \mathbf{p}, \mathbf{r}) \, d\mu^*(\mathbf{p}, \mathbf{r}) = \int u(\pi^*, \mathbf{p}, \mathbf{r}) \, d\mu^*(\mathbf{p}, \mathbf{r}) = \inf_{\mu \in \mathcal{C}_S} \int u(\pi^*, \mathbf{p}, \mathbf{r}) \, d\mu(\mathbf{p}, \mathbf{r}).$$

Therefore, to find the sequentially robust policy, we need only to solve the sequentially robust action.

**Theorem 2.** *Denote $\lambda_0 = 0$. For $s \in S_t$ where $t < T$, the sequentially robust action is given by*

$$\mathbf{q}^* = \arg \max_{\mathbf{q} \in \Delta(s)} \left\{ \sum_{i=1}^n (\lambda_i - \lambda_{i-1}) \min_{(\mathbf{p}_s^i, \mathbf{r}_s^i) \in \mathcal{P}_s^i} \left[(\mathbf{r}_s^i)^\top \mathbf{q} + (\mathbf{p}_s^i)^\top \tilde{V}_s \mathbf{q}\right] \right\}, \tag{2}$$

*where $m = |A_s|$, $\tilde{\mathbf{v}}_{t+1}$ is the vector form of $\tilde{v}_{t+1}(s')$ for all $s' \in S_{t+1}$, and*

$$\tilde{V}_s \triangleq \begin{bmatrix} \tilde{\mathbf{v}}_{t+1} \mathbf{e}_1^\top(m) \\ \vdots \\ \tilde{\mathbf{v}}_{t+1} \mathbf{e}_m^\top(m) \end{bmatrix}.$$

Theorem 2 implies that the computation of the sequentially robust action at a state $s$ critically depends on the structure of the sets $\mathcal{P}_s^i$. In fact, it can be shown that for "good" uncertainty sets, computing the sequentially robust action is tractable. This claim is made precise by the following corollary. We omit the proof that is standard.

**Corollary 1.** *The sequentially robust action for state $s$ can be found in polynomial-time, if for each $i = 1, \cdots, n$, $\mathcal{P}_s^i$ has a polynomial separation oracle. Here, a polynomial separation oracle of a convex set $\mathcal{H} \subseteq \mathbb{R}^n$ is a subroutine that given $\mathbf{x} \in \mathbb{R}^n$, reports in polynomial time whether $\mathbf{x} \in \mathcal{H}$, and if the answer is negative, it finds a hyperplane that separates $\mathbf{x}$ and $\mathcal{H}$.*

### 3.1 Proof of Theorem 1

We prove Theorem 1 in this section. The outline of the proof is as follows: We first show that for a given policy, the expected performance under an admissible $\mu$ depends only on the expected value of the parameters. Then we show that the set of expected parameters is indeed $\bigotimes_{s \in S} \hat{\mathcal{P}}_s$. Thus the

distributionally robust MDP reduces to the robust MDP with $\bigotimes_{s \in S} \hat{\mathcal{P}}_s$ being the uncertainty set. Finally, by applying results from robust MDPs we prove the theorem. Some of the intermediate results are stated with proof omitted due to space constraints.

Let $h_t$ denote a history up to stage $t$ and $s(h_t)$ denote the last state of history $h_t$. We use $\pi_{h_t}(a)$ to represent the probability of choosing an action $a$ at state $s(h_t)$, following a policy $\pi$ and under a history $h_t$. A $t+1$ stage history, with $h_t$ followed by action $a$ and state $s'$ is written as $(h_t, a, s')$. With an abuse of notation, we denote the expected reward-to-go under a history as:

$$u(\pi, \mathbf{p}, \mathbf{r}, h_t) \triangleq \mathbb{E}_\pi^{\mathbf{P}}\{\sum_{i=t}^T \gamma^{i-t} r(s_i, a_i) | (s_1, a_1 \cdots, s_t) = h_t\}.$$

For $\pi \in \Pi^{HR}$ and $\mu \in \mathcal{C}_S(\lambda)$, define $w(\pi, \mu, h_t) \triangleq \mathbb{E}_{(\mathbf{p}, \mathbf{r}) \sim \mu} u_s(\pi, \mathbf{p}, \mathbf{r}, h(t)) = \int u(\pi, \mathbf{p}, \mathbf{r}, h(t)) d\mu(\mathbf{p}, \mathbf{r})$. Thus, $w(\pi, \mu, (s^{\text{ini}})) = \int u(\pi, \mathbf{p}, \mathbf{r}) \, d\mu(\mathbf{p}, \mathbf{r})$ is the minimax objective. One can show that the following recursion formula for $w(\cdot)$ holds, due to the fact that $\mu(\mathbf{p}, \mathbf{r}) = \bigotimes_{s \in S} \mu_s(\mathbf{p}_s, \mathbf{r}_s)$.

**Lemma 1.** Fix $\pi \in \Pi^{HR}$, $\mu \in \mathcal{C}_S$ and a history $h_t$ where $t < T$, denote $\overline{\mathbf{r}} = \mathbb{E}_\mu(\mathbf{r})$, $\overline{\mathbf{p}} = \mathbb{E}_\mu(\mathbf{p})$, then we have:

$$w(\pi, \mu, h_t) = \int \sum_{a \in A_{s(h_t)}} \pi_{h_t}(a) \Big( r\big(s(h_t), a\big) + \sum_{s' \in S} \gamma p(s'|s(h_t), a) w\big(\pi, \mu, (h_t, a, s')\big) \Big) d\mu_{s(h_t)}(\mathbf{p}_{s(h_t)}, \mathbf{r}_{s(h_t)})$$

$$= \sum_{a \in A_{s(h_t)}} \pi_{h_t}(a) \Big( \overline{r}\big(s(h_t), a\big) + \sum_{s' \in S} \gamma \overline{p}(s'|s(h_t), a) w\big(\pi, \mu, (h_t, a, s')\big) \Big).$$

From Lemma 1, by backward induction, one can show the following lemma holds, which essentially means that for any policy, the expected performance under an admissible distribution $\mu$ only depends on the expected value of the parameters under $\mu$. Thus, the distributionally robust MDP reduces to a robust MDP.

**Lemma 2.** Fix $\pi \in \Pi^{HR}$ and $\mu \in \mathcal{C}_S$, denote $\overline{\mathbf{p}} = \mathbb{E}_\mu(\mathbf{p})$ and $\overline{\mathbf{r}} = \mathbb{E}_\mu(\mathbf{r})$. We have: $w\big(\pi, \mu, (s^{\text{ini}})\big) = u(\pi, \overline{\mathbf{p}}, \overline{\mathbf{r}})$.

Next we characterize the set of expected value of the parameters.

**Lemma 3.** Fix $s \in S$, we have $\{\mathbb{E}_{\mu_s}(\mathbf{p}_s, \mathbf{r}_s) | \mu_s \in \mathcal{C}_s\} = \hat{\mathcal{P}}_s$.

Note that Lemma 3 implies that $\{\mathbb{E}_\mu(\mathbf{p}, \mathbf{r}) | \mu \in \mathcal{C}_S\} = \bigotimes_{s \in S} \hat{\mathcal{P}}_s$. We complete the proof of the Theorem 1 using the equivalence of distributionally robust MDPs and robust MDPs where the uncertainty set is $\bigotimes_{s \in S} \hat{\mathcal{P}}_s$. Recall that for each $s \in S$, $\hat{\mathcal{P}}_s$ is convex and compact. It is well known that for robust MDPs, a saddle point of the minimax objective exists (cf [5, 8]). More precisely, there exists $\pi^* \in \Pi^{HR}$, $(\mathbf{p}^*, \mathbf{r}^*) \in \bigotimes_{s \in S} \hat{\mathcal{P}}_s$ such that

$$\sup_{\pi \in \Pi^{HR}} u(\pi, \mathbf{p}^*, \mathbf{r}^*) = u(\pi^*, \mathbf{p}^*, \mathbf{r}^*) = \inf_{(\mathbf{r}, \mathbf{p}) \in \bigotimes_{s \in S} \hat{\mathcal{P}}_s} u(\pi^*, \mathbf{p}, \mathbf{r}).$$

Moreover, $\pi^*$ and $(\mathbf{p}^*, \mathbf{r}^*)$ can be constructed state-wise: $\pi^* = \bigotimes_{s \in S} \pi_s^*$ and $(\mathbf{p}^*, \mathbf{r}^*) = \bigotimes_{s \in S}(\mathbf{p}_s^*, \mathbf{r}_s^*)$, and for each $s \in S_t$, $\pi_s^*, (\mathbf{p}_s^*, \mathbf{r}_s^*)$ solves the following zero-sum game

$$\max_{\pi_s} \min_{(\mathbf{p}_s, \mathbf{r}_s) \in \hat{P}_s} \mathbb{E}_\pi^{\mathbf{P}_s}\big(r(s, a) + \gamma \tilde{v}_{t+1}(\underline{s})\big).$$

It follows that $\pi_s^*$ is any sequentially robust action, and hence $\pi^*$ can be any sequentially robust policy. From Lemma 3, there exists $\mu_s^* \in \mathcal{C}_s$ that satisfies $\mathbb{E}_{\mu_s^*}(\mathbf{p}_s, \mathbf{r}_s) = (\mathbf{p}_s^*, \mathbf{r}_s^*)$. Let $\mu^* = \bigotimes_{s \in S} \mu_s^*$. By Lemma 2 we have

$$\sup_{\pi \in \Pi^{HR}} w\big(\pi, \mu^*, (s^{\text{ini}})\big) = \sup_{\pi \in \Pi^{HR}} u(\pi, \mathbf{p}^*, \mathbf{r}^*);$$

$$w\big(\pi^*, \mu^*, (s^{\text{ini}})\big) = u\big(\pi^*, \mathbf{p}^*, \mathbf{r}^*\big);$$

$$\inf_{\mu \in \mathcal{C}_S} w\big(\pi^*, \mu, (s^{\text{ini}})\big) = \inf_{(\mathbf{p}, \mathbf{r}) \in \bigotimes_s \hat{\mathcal{P}}_s} u(\pi^*, \mathbf{p}, \mathbf{r}).$$

This leads to $\sup_{\pi\in\Pi^{HR}} w\big(\pi,\mu^*,(s^{\text{ini}})\big) = w\big(\pi^*,\mu^*,(s^{\text{ini}})\big) = \inf_{\mu\in\mathcal{C}_S} w\big(\pi^*,\mu,(s^{\text{ini}})\big)$. Thus, part (ii) of Theorem 1 holds. Note that part (ii) immediately implies part (i) of Theorem 1.

**Remark:** Lemma 1 holds for a broader class of distribution sets than we discussed here. Indeed, the only requirement of $\mathcal{C}$ for Lemma 1 to hold is the state-wise decomposibility. Therefore, the results presented in this paper may well extend to distributionally robust MDPs whose parameters belongs to other interesting sets of distributions, such as a set of parametric distribution (Gaussian, exponential, binomial etc) with the distribution parameter not precisely determined.

## 4   Distributionally robust MDP: The discounted reward infinite horizon case.

In this section we show how to compute a distributionally robust policy for infinite horizon MDPs. Specifically, we generalize the notion of sequentially robust policies to discounted-reward infinite-horizon MDPs, and show that it is distributionally robust in an appropriate sense.

**Definition 3.** *Let* $T = \infty$ *and* $\gamma < 1$. *Denote the uncertainty set by* $\hat{\mathcal{P}} = \bigotimes_s \hat{\mathcal{P}}_s$. *We define the following:*

1. *The* sequentially robust value $\tilde{v}_\infty(s)$ *w.r.t.* $\hat{P}_s$ *is the unique solution to the following set of equations:*

$$\tilde{v}_\infty(s) = \max_{\pi_\mathbf{s}\in\Delta(s)}\left\{\min_{(\mathbf{p}_s,\mathbf{r}_s)\in\hat{\mathcal{P}}_s}\mathbb{E}^{\mathbf{P}_s}_{\pi_s}[r(s,a)+\gamma\tilde{v}_\infty(\underline{s})]\right\}, \forall s\in S.$$

2. *The* sequentially robust action *w.r.t.* $\hat{P}_s$, $\tilde{\pi}_s$, *is given by*

$$\tilde{\pi}_s \in \arg\max_{\pi_\mathbf{s}\in\Delta(s)}\left\{\min_{(\mathbf{p}_s,\mathbf{r}_s)\in\hat{\mathcal{P}}_s}\mathbb{E}^{\mathbf{P}_s}_{\pi_s}[r(s,a)+\gamma\tilde{v}_\infty(\underline{s})]\right\}.$$

3. *A policy* $\tilde{\pi}^*$ *is a* sequentially robust policy *w.r.t.* $\hat{P}_s$ *if* $\forall s\in S$, $\tilde{\pi}_s^*$ *is a sequentially robust action.*

The sequentially robust policy is well defined, since the following operator $\mathcal{L}:\mathbb{R}^{|S|}\to\mathbb{R}^{|S|}$ is a $\gamma$ contraction for $\|\cdot\|_\infty$ norm.

$$\{\mathcal{L}\mathbf{v}\}(s) \triangleq \max_{\mathbf{q}\in\Delta(s)}\min_{(\mathbf{p},\mathbf{r})\in\hat{\mathcal{P}}_s}\Big[\sum_{a\in A_s}q(a)r(s,a)+\gamma\sum_{a\in A_s}\sum_{s'\in S}q(a)p(s'|s,a)v(s')\Big].$$

Furthermore, given any $\mathbf{v}$, applying $\mathcal{L}$ is equivalent to solving a minimax problem, which by Theorem 2 can be efficiently computed. Hence, by applying $\mathcal{L}$ on any initial $\mathbf{v}^0\in\mathbb{R}^{|S|}$ repeatedly, the resulting value vector will converge to the sequentially robust value $\tilde{\mathbf{v}}$ exponentially fast.

Note that in the infinite horizon case, we cannot model the system as (1) having finitely many states, and (2) each visited at most once. In contrast, we have to relax either one of these two assumptions, leading to two different natural formulations. The first formulation, termed *non-stationary model*, is to treat the system as having infinitely many states, each visited at most once. Therefore, we consider an equivalent MDP with an augmented state space, where each augmented state is defined by a pair $(s,t)$ where $s\in S$ and $t$, meaning state $s$ in the $t^{th}$ horizon. Thus, each augmented state will be visited at most once, which leads to the following set of distributions.

$$\bar{\mathcal{C}}_S^\infty \triangleq \{\mu|\mu = \bigotimes_{s\in S, t=1,2,\cdots} \mu_{s,t}; \ \mu_{s,t}\in\mathcal{C}_s, \forall s\in S, \forall t=1,2,\cdots\}.$$

The second formulation, termed *stationary model*, treats the system as having a finite number of states, while multiple visits to one state is allowed. That is, if a state $s$ is visited for multiple times, then each time the distribution (of uncertain parameters) $\mu_s$ is the same. Mathematically, we can adapt the augmented state space as in the non-stationary model, and requires that $\mu_{s,t}$ does not depend on $t$. Thus, the set of admissible distributions is

$$\bar{\mathcal{C}}_S \triangleq \{\mu|\mu = \bigotimes_{s\in S, t=1,2,\cdots} \mu_{s,t}; \ \mu_{s,t}=\mu_s; \mu_s\in\mathcal{C}_s, \forall s\in S, \forall t=1,2,\cdots\}.$$

The next theorem is the main result of this section; it shows that a sequentially robust policy is distributionally robust to both stationary and non-stationary models.

**Theorem 3.** *Given $T = \infty$ and $\gamma < 1$, any sequentially robust policy w.r.t. $\bigotimes_s \hat{\mathcal{P}}_s$ where $\hat{\mathcal{P}}_s = \{\sum_{i=1}^n (\lambda_i - \lambda_{i-1})(\mathbf{r}_s(i), \mathbf{p}_s(i)) | (\mathbf{p}_s(i), \mathbf{r}_s(i)) \in \mathcal{P}_s^i\}$, is distributionally robust with respect to $\bar{\mathcal{C}}_S^\infty$, and with respect to $\bar{\mathcal{C}}_S$.*

Due to space constraints, we omit the proof details. The basic idea for proving the $\bar{\mathcal{C}}_S^\infty$ case is to consider a $\hat{T}$-truncated problem, i.e., a finite horizon problem that stops at stage $\hat{T}$ with a termination reward $\tilde{v}_\infty(\cdot)$, and show that the optimal strategy for this problem, which is a sequential robust strategy, coincides with that of the infinite horizon one. Indeed, given any sequential robust strategy $\pi^*$, one can construct a stationary distribution $\mu^*$ such that $(\pi^*, \mu^*)$ is a saddle point for $\sup_{\pi \in \Pi^{HR}} \inf_{\mu \in \bar{\mathcal{C}}_S^\infty} \int u(\pi, \mathbf{p}, \mathbf{r}) \, d\mu(\mathbf{p}, \mathbf{r})$. The proof to $\bar{\mathcal{C}}_S$ follows from $\bar{\mathcal{C}}_S \subset \bar{\mathcal{C}}_S^\infty$ and $\mu^* \in \bar{\mathcal{C}}_S$. We remark that the decision maker is allowed to take non-stationary strategies, although the distributionally robust solution is proven to be stationary.

Before concluding this section, we briefly compare the stationary model and the non-stationary model. These two formulations model different setups: if the system, more specifically the distribution of uncertain parameters, evolves with time, then the non-stationary model is more appropriate; while if the system is static, then the stationary model is preferable. For any given policy, the worst expected performance under the non-stationary model provides a lower bound to that of the stationary model since $\bar{\mathcal{C}}_S \subseteq \bar{\mathcal{C}}_S^\infty$. Thus, one can use the non-stationary model to approximate the stationary model, when the latter is intractable (e.g., in the finite horizon case; see Nilim and El Ghaoui [5]). When the horizon approaches infinity, such approximation becomes exact, as we showed in this section, the optimal solutions to both formulations coincide, and can be computed by iteratively solving a minimax problem.

## 5 Numerical simulations

In this section we illustrate with numerical examples that by incorporating additional probabilistic information, the distributional robust approach handles uncertainty in a more flexible way, which often leads to a better performance than the nominal approach and the robust approach.

We consider a path planning problem: an agent wants to exit a $4 \times 21$ maze (shown in Figure 1) using the least possible time. Starting from the upper-left corner, the agent can move up, down, left and right, but can only exit the grid at the lower-right corner. Here, a white box stands for a normal place where the agent needs one time unit to pass through. A shaded box represents a "shaky" place. To be more specific, we consider two setups. The first one is the uncertain cost case, where the true (yet unknown to the planning agent) time for the agent to pass through a "shaky" place equals $x = 1 + \tilde{e}(\lambda)$, and $\tilde{e}(\lambda)$ is an exponential distributed random variable with parameter $\lambda$. The three approaches are formulated as follows: the nominal approach takes the most likely value (i.e., 1) as the parameter; the robust approach takes $[1, 1 + 3/\lambda]$ as the uncertainty set; and the distributional robust approach takes into account the additional information that $\Pr(x \in [1, 1 + \log 2/\lambda]) \geq 0.5$ and $\Pr(x \in [1, 1 + 2\log 2/\lambda]) \geq 0.75$. We vary $1/\lambda$, and test these approaches using 300 runs for each parameter set. The results are reported in Figure 2 (a).

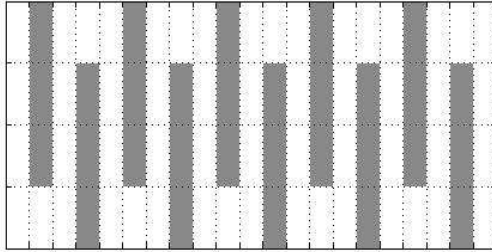

Figure 1: The maze for the path planning problem.

The second case is the uncertain transition case: if an agent reaches a "shaky" place, then the transition becomes unpredictable – in the next step with probability $20\%$ it will make an (unknown)

jump. The three approaches are set as follows: The nominal approach neglects this random jump. The robust approach takes a worst-case analysis, i.e., it assumes that with $20\%$ the agent will jump to the spot with the highest cost-to-go. The distributionally robust approach takes into account an additional information that if a jump happens, the probability that it jumps to a spot that is left to the current place is no more than $\gamma$. Each policy is tested over $300$ runs, while the true jump is set as with probability $0.2\gamma$ the agent returns to the starting point ("reboot"), with $0.2(1-\gamma)$ the agent stay in the current position for a time unit ("stuck"). The results are reported in Figure 2 (b).

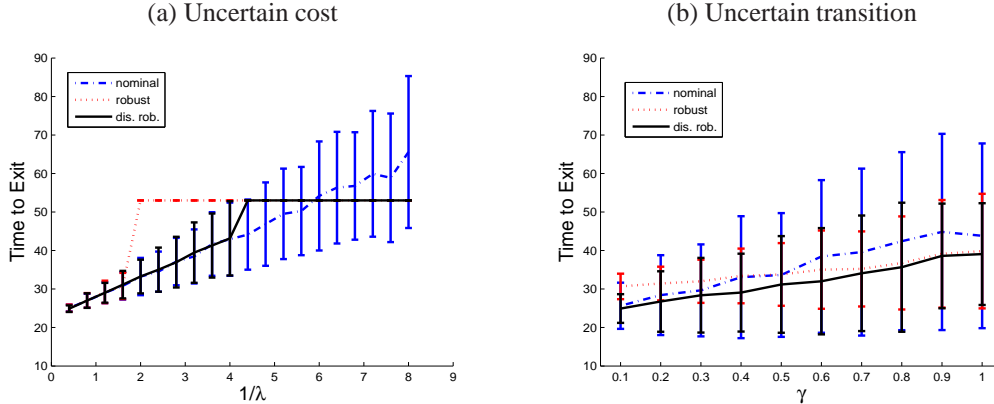

Figure 2: Simulation results of the path planning problem.

In both the uncertain cost and the uncertain transition probability setups, the distributionally robust approach outperforms the other two approach over virtually the whole range of parameters. This is well expected, since additional probabilistic information is available to and incorporated by the distributionally robust approach.

## 6   Concluding remarks

In this paper we proposed a distributionally robust approach to mitigate the conservatism of the robust MDP framework and incorporate additional a-prior probabilistic information regarding the unknown parameters. In particular, we considered the nested-set structured parameter uncertainty to model a-prior probabilistic information of the parameters. We proposed to find a policy that achieves maximum expected utility under the worst admissible distribution of the parameters. Such formulation leads to a policy that is obtained through a Bellman type backward induction, and can be solved in polynomial time under mild technical conditions.

A different perspective on our work is that we develop a principled approach to the problem of uncertainty set design in multi-stage decision problems. It has been observed that shrinking the uncertainty set in single-stage problems leads to better performance. We provide a principled approach to the problem of uncertainty set selection: the distributionally robust policy is a robust policy w.r.t. a carefully designed single uncertainty set that depends on the a-priori knowledge.

A natural question is how can we take advantage of the distributionally robust approach and solve (exactly) a full-blown Bayesian generative model MDP? The problem with taking an increasingly refined nested uncertainty structure (i.e., increasing $n$) is that of representation: the equivalent robust MDP uncertainty set may become too complicated to represent efficiently. Nevertheless, if it is possible to offer upper and lower bounds on the probability of each nested sets (based on the generative model), the corresponding distributionally robust policies provide performance bounds on the optimal policies in the, often intractable, Bayesian model.

#### Acknowledgements

We thank an anonymous reviewer for pointing out relevant references in mathematical finance. H. Xu would like to acknowledge the support from DTRA grant HDTRA1-08-0029. S. Mannor would like to acknowledge the support the Israel Science Foundation under contract 890015.

# References

[1] M. L. Puterman. *Markov Decision Processes*. John Wiley & Sons, New York, 1994.

[2] D. P. Bertsekas and J. N. Tsitsiklis. *Neuro-Dynamic Programming*. Athena Scientific, 1996.

[3] R. S. Sutton and A. G. Barto. *Reinforcement Learning: An Introduction*. MIT Press, 1998.

[4] S. Mannor, D. Simester, P. Sun, and J. Tsitsiklis. Bias and variance in value vunction estimation. In *Proceedings of the 21th international conference on Machine learning*, 2004.

[5] A. Nilim and L. El Ghaoui. Robust control of Markov decision processes with uncertain transition matrices. *Operations Research*, 53(5):780–798, September 2005.

[6] A. Bagnell, A. Ng, and J. Schneider. Solving uncertain Markov decision problems. Technical Report CMU-RI-TR-01-25, Carnegie Mellon University, August 2001.

[7] C. C. White III and H. K. El Deib. Markov decision processes with imprecise transition probabilities. *Operations Research*, 42(4):739–748, July 1992.

[8] G. N. Iyengar. Robust dynamic programming. *Mathematics of Operations Research*, 30(2):257–280, 2005.

[9] L. G. Epstein and M. Schneider. Learning under ambiguity. *Review of Economic Studies*, 74(4):1275–1303, 2007.

[10] A. Nilim and L. El Ghaoui. Robustness in Markov decision problems with uncertain transition matrices. In *Advances in Neural Information Processing Systems 16*, 2004.

[11] E. Delage and S. Mannor. Percentile optimization for Markov decision processes with parameter uncertainty. *Operations Research*, (1):203–213, 2010.

[12] H. Xu and S. Mannor. The robustness-performance tradeoff in Markov decision processes. In B. Schölkopf, J. C. Platt, and T. Hofmann, editors, *Advances in Neural Information Processing Systems 19*, pages 1537–1544. MIT Press, 2007.

[13] H. Scarf. A min-max solution of an inventory problem. In *Studies in Mathematical Theory of Inventory and Production*, pages 201–209. Stanford University Press, 1958.

[14] J. Dupacová. The minimax approach to stochastic programming and an illustrative application. *Stochastics*, 20:73–88, 1987.

[15] P. Kall. Stochastic programming with recourse: Upper bounds and moment problems, a review. In *Advances in Mathematical Optimization*. Academie-Verlag, Berlin, 1988.

[16] A. Shapiro. Worst-case distribution analysis of stochastic programs. *Mathematical Programming*, 107(1):91–96, 2006.

[17] I. Popescu. Robust mean-covariance solutions for stochastic optimization. *Operations Research*, 55(1):98–112, 2007.

[18] E. Delage and Y. Ye. Distributional robust optimization under moment uncertainty with applications to data-driven problems. To appear in *Operations Research*, 2010.

[19] A. Ruszczyński. Risk-averse dynamic programming for Markov decision processes. *Mathematical Programming, Series B*, 125:235–261, 2010.

[20] H. Föllmer and A. Schied. *Stochastic finance: An introduction in discrete time*. Berlin: Walter de Gruyter, 2002.

[21] I. Gilboa and D. Schmeidler. Maxmin expected utility with a non-unique prior. *Journal of Mathematical Economics*, 18:141–153, 1989.

[22] D. Kelsey. Maxmin expected utility and weight of evidence. *Oxford Economic Papers*, 46:425–444, 1994.

[23] J. Baron. *Thinking and Deciding*. Cambridge University Press, 2000.

